# Learning Segmentation by Random Walks

**Marina Meilă**
University of Washington
mmp@stat.washington.edu

**Jianbo Shi**
Carnegie Mellon University
jshi@cs.cmu.edu

## Abstract

We present a new view of image segmentation by pairwise similarities. We interpret the similarities as edge flows in a Markov random walk and study the eigenvalues and eigenvectors of the walk's transition matrix. This interpretation shows that spectral methods for clustering and segmentation have a probabilistic foundation. In particular, we prove that the Normalized Cut method arises naturally from our framework. Finally, the framework provides a principled method for learning the similarity function as a combination of features.

## 1 Introduction

This paper focuses on *pairwise* (or *similarity-based*) clustering and image segmentation. In contrast to statistical clustering methods, that assume a probabilistic model that generates the observed data points (or pixels), pairwise clustering defines a *similarity function* between pairs of points and then formulates a criterion (e.g. maximum total intracluster similarity) that the clustering must optimize. The optimality criteria quantify the intuitive notion that points in a cluster (or pixels in a segment) are similar, whereas points in different clusters are dissimilar.

An increasingly popular approach to similarity based clustering and segmentation is by spectral methods. These methods use eigenvalues and eigenvectors of a matrix constructed from the pairwise similarity function. Spectral methods are sometimes regarded as continuous approximations of previously formulated discrete graph theoretical criteria as in image segmentation method of [9], or as in the web clustering method of [4, 2]. As demonstrated in [9, 4], these methods are capable of delivering impressive segmentation/clustering results using simple low-level features.

In spite of their practical successes, spectral methods are still incompletely understood. The main achievement of this work is to show that there is a simple probabilistic interpretation that can offer insights and serve as an analysis tool for all the spectral methods cited above. We view the pairwise similarities as edge flows in a Markov random walk and study the properties of the eigenvectors and values of the resulting transition matrix. Using this view, we were able to show that several of the above methods are subsumed by the Normalized Cut (NCut) segmentation algorithm of [9] in a sense that will be described. Therefore, in the following, we will focus on the NCut algorithm and will adopt the terminology of image segmentation (i.e. the data points are *pixels* and the set of all pixels is the *image*), keeping in mind that all the results are also valid for similarity based clustering.

A probabilistic interpretation of NCut as a Markov random walk not only sheds new lights on why and how spectral methods work in segmentation, but also offers a principled way of learning the similarity function. A segmented image can provide a "target" transition matrix to which a learning algorithm matches in KL divergence the "learned" transition probabilities. The latter are output by a model as a function of a set of features measured from the training image. This is described in section 5. Experimental results on learning segmenting objects with smooth and rounded shape is described in section 6.

## 2 The Normalized Cut criterion and algorithm

Here and in the following, an image will be represented by a set of pixels $I$. A segmentation is a partitioning of $I$ into mutually disjoint subsets. For each pair of pixels $i, j \in I$ a similarity $S_{ij} = S_{ji} \geq 0$ is given. In the NCut framework the similarities $S_{ij}$ are viewed as weights on the edges $ij$ of a graph $G$ over $I$. The matrix $S = [S_{ij}]$ plays the role of a "real-valued" adjacency matrix for $G$. Let $d_i = \sum_{j \in I} S_{ij}$, called the *degree* of node $i$, and the *volume* of a set $A \subset I$ be $\text{vol } A = \sum_{i \in A} d_i$. The set of edges between $A$ and its complement $\bar{A}$ is an *edge cut* or shortly a *cut*. The *normalized cut* (NCut) criterion of [9] is a graph theoretical criterion for segmenting an image into two by minimizing

$$NCut(A, \bar{A}) = \left( \frac{1}{\text{vol } A} + \frac{1}{\text{vol } \bar{A}} \right) \sum_{i \in A, j \in \bar{A}} S_{ij} \qquad (1)$$

over all cuts $A, \bar{A}$. Minimizing $NCut$ means finding a cut of relatively small weight between two subsets with strong internal connections. In [9] it is shown that optimizing $NCut$ is NP hard.

The *NCut algorithm* was introduced in [9] as a continuous approximation for solving the discrete minimum NCut problem by way of eigenvalues and eigenvectors. It uses the *Laplacian* matrix $L = D - S$ where $D$ is a diagonal matrix formed with the degrees of the nodes. The algorithm consists of solving the generalized eigenvalues/vectors problem

$$Lx = \lambda Dx \qquad (2)$$

The NCut algorithm focuses on the second smallest eigenvalue of (2) and its corresponding eigenvector, call them $\lambda^L$ and $x^L$ respectively. In [9] it is shown that when there is a partitioning of $A, \bar{A}$ of $I$ such that

$$x_i^L = \begin{cases} \alpha, & i \in A \\ \beta, & i \in \bar{A} \end{cases} \qquad (3)$$

then $A, \bar{A}$ is the optimal NCut and the value of the cut itself is $NCut(A, \bar{A}) = \lambda^L$.

This result represents the basis of spectral segmentation by normalized cuts. One solves the generalized spectral problem (2), then finds a partitioning of the elements of $x^L$ into two sets containing roughly equal values. The partitioning can be done by thresholding the elements. The partitioning of the eigenvector induces a partition on $I$ which is the desired segmentation.

As presented above, the NCut algorithm lacks a satisfactory intuitive explanation. In particular, the NCut algorithm and criterion offer little intuition about (1) what causes $x^L$ to be piecewise constant? (2) what happens when there are more than two segments and (3) how does the algorithm degrade its performance when $x^L$ is not piecewise constant?

The random walk interpretation that we describe now will answer the first two questions as well as give a better understanding of what spectral clustering is achieving. We shall not approach the third issue here: instead, we point to the results of [2] that apply to the NCut algorithm as well.

## 3   Markov walks and normalized cuts

By "normalizing" the similarity matrix $S$ one obtains the stochastic matrix

$$P = D^{-1}S \tag{4}$$

whose row sums are all 1. As it is known from the theory of Markov random walks, $P_{ij}$ represents the probability of moving from node $i$ to $j$ in one step, given that we are in $i$. The eigenvalues of $P$ are $\lambda_1 = 1 \geq \lambda_2 \geq \ldots \lambda_n \geq -1$; $x^{1\ldots n}$ are the eigenvectors. The first eigenvector of $P$ is $x^1 = \mathbf{1}$, the vector whose elements are all 1s. W.l.o.g we assume that no node has degree 0.

Let us now examine the spectral problem for the matrix $P$, namely the solutions of the equation

$$Px = \lambda x \tag{5}$$

**Proposition 1** *If $\lambda$, $x$ are solutions of (5) and $P = D^{-1}S$, then $(1 - \lambda)$, $x$ are solutions of (2).*

In other words, the NCut algorithm and the matrix $P$ have the same eigenvectors; the eigenvalues of $P$ are identical to 1 minues the generalized eigenvalues in (2). Proposition 1 shows the equivalence between the spectral problem formulated by the NCut algorithm and the eigenvalues/vectors of the stochastic matrix $P$. This also helps explaining why the NCut algorithm uses the second smallest generalized eigenvector: the smallest eigenvector of (2) corresponds to the largest eigenvector of $P$, which in most cases of interest is equal to $\mathbf{1}$ thus containing no information.

The NCut criterion can also be understood in this framework. First define $\pi^\infty = [\pi_i^\infty]_{i \in I}$ by $\pi_i^\infty = \frac{d_i}{\mathrm{vol}I}$. It is easy to verify that $P^T \pi^\infty = \pi^\infty$ and thus that $\pi^\infty$ is a *stationary distribution* of the Markov chain. If the chain is ergodic, which happens under mild conditions [1], then $\pi^\infty$ is the only distribution over $I$ with this property. Note also that the Markov chain is *reversible* because $\pi_i^\infty P_{ij} = \pi_j^\infty P_{ji} = S_{ij}/\mathrm{vol}I$. Define $P_{AB} = Pr[A \to B|A]$ as the probability of the random walk transitioning from set $A \subset I$ to set $B \subset I$ in one step if the current state is in $A$ and the random walk is started in its stationary distribution.

$$P_{AB} = \frac{\sum_{i \in A, j \in B} \pi_i^\infty P_{ij}}{\pi^\infty(A)} = \frac{\sum_{i \in A, j \in B} S_{ij}}{\mathrm{vol}(A)} \tag{6}$$

From this it follows that

$$NCut(A, \bar{A}) = P_{A\bar{A}} + P_{\bar{A}A} \tag{7}$$

If the NCut is small for a certain partition $A, \bar{A}$ then it means that the probabilities of evading set $A$, once the walk is in it and of evading its complement $\bar{A}$ are both small. Intuitively, we have partitioned the set $I$ into two parts such that the random walk, once in one of the parts, tends to remain in it.

The NCut is strongly related to a the concept of low conductivity sets in a Markov random walk. A *low conductivity set* $A$ is a subset of $I$ such that $h(A) = \max(P_{A\bar{A}}, P_{\bar{A}A})$ is small. They have been studied in spectral graph theory in connection with the *mixing time* of Markov random walks [1]. More recently, [2] uses them to define a new criterion for clustering. Not coincidentally, the heuristic analyzed there is strongly similar to the NCut algorithm.

# 4 Stochastic matrices with piecewise constant eigenvectors

In the following we will use the transition matrix $P$ to achieve a better understanding of the NCut algorithm. Recall that the NCut algorithm looks at the second "largest" eigenvector of $P$, denoted by $x^2$ and equal to $x^L$, in order to obtain a partioning of $I$. We define a vector $x$ to be *piecewise constant* relative to a partition $\Delta = (A_1, A_2, \ldots A_k)$ of $I$ iff $x_i = x_j$ for $i, j$ pixels in the same set $A_s$, $s = 1, \ldots k$. Since having piecewise constant eigenvectors is ideal case for spectral segmentation, it is important to understand when the matrix $P$ has this desired property. We study when the first $k$ out of $n$ eigenvectors are piecewise constant.

**Proposition 2** *Let $P$ be a matrix with rows and columns indexed by $I$ that has independent eigenvectors. Let $\Delta = (A_1, A_2, \ldots A_k)$ be a partition of $I$. Then, $P$ has $k$ eigenvectors that are piecewise constant w.r.t. $\Delta$ and correspond to non-zero eigenvalues if and only if the sums $P_{is} = \sum_{j \in A_s} P_{ij}$ are constant for all $i \in A_{s'}$ and all $s, s' = 1, \ldots k$ and the matrix $R = [P_{ss'}]_{s,s'=1,\ldots k}$ (with $P_{ss'} = \sum_{j \in A'_s} P_{ij}$, $i \in A_s$) is non-singular.*

**Lemma 3** *If the matrix $P$ of dimension $n$ is of the form $P = D^{-1}S$ with $S$ symmetric and $D$ non-singular then $P$ has $n$ independent eigenvectors.*

We call a stochastic matrix $P$ satisfying the conditions of Proposition 2 a block-stochastic matrix. Intuitively, Proposition 2 says that a stochastic matrix has piecewise constant eigenvectors if the underlying Markov chain can be aggregated into a Markov chain with state space $\Delta = \{A_1, \ldots A_k\}$ and transition probability matrix $\hat{P}$. This opens interesting connections between the field of spectral segmentation and the body of work on aggregability or (*lumpability*) [3] of Markov chains. The proof of Proposition 2 is provided in [5].

Proposition 2 shows that a much broader condition exists for Ncut algorithm to produce an exact segmentation/clustering solution. Such condition shows that in fact *spectral clustering is able to group pixels by the similarity of their transition probabilities to subsets of $I$.* Experiments [9] show that NCut works well on many graphs that have a sparse complex connection structure supporting this result with practical evidence. Proposition 2 generalizes previous results of [10].

The NCut algorithm and criterion is one of the recently proposed spectral segmentation methods. In image segmentation, there are algorithms of Perona and Freeman (PF) [7] and Scott and Longuet-Higgins (SLH) [8]. In web clustering, there are algorithms of Kleinberg[4](K), the long known latent semantic analysis (LSA), and in the variant proposed by Kannan, Vempala and Vetta (KVV) [2]. It is easy to show that each of the above ideal situations imply that the resulting stochastic matrix $P$ satisfies the conditions of Proposition 2 and thus the NCut algorithm will also work exactly in these situations. In this sense NCut subsumes PF, SLH and (certain variants of) K. Moreover, none of the three other methods takes into account more information than NCut does. Another important aspect of a spectral clustering algorithm is robustness. Empirical results of [10] show that NCut is at least as robust as PF and SLH.

# 5 The framework for learning image segmentation

The previous section stressed the connection between NCut as a criterion for image segmentation and searching for low conductivity sets in a random walk. Here we will exploit this connection to develop a framework for supervised learning of image segmentation. Our goal is to obtain an algorithm that starts with a training set of

segmented images and with a set of features and learns a function of the features that produces correct segmentations, as shown in figure 1.

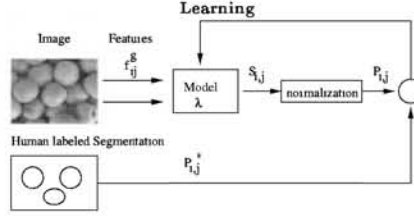

Figure 1: The general framework for learning image segmentation.

For simplicity, assume the training set consists of one image only and its correct segmentation. From the latter it is easy to obtain "ideal" or *target* transition probabilities

$$P_{ij}^* = \begin{cases} 0, & j \notin A \\ \frac{1}{|A|}, & j \in A. \end{cases} \quad \text{for } i \text{ in segment } A \text{ with } |A| \text{ elements} \quad (8)$$

We also have a predefined set of features $f^q$, $q = 1, \ldots Q$ which measure similarity between two pixels according to different criteria and their values for I.

The *model* is the part of the framework that is subject to learning. It takes the features $f_{ij}^q$ as inputs and outputs the global similarity measure $S_{ij}$. For the present experiments we use the simple model $S_{ij} = e^{\sum_q \lambda_q f_{ij}^q}$ Intuitively, it represents a set of independent "experts", the factors $e^{\lambda_q f^q}$ voting on the probability of a transition $i \to j$.

In our framework, based on the fact that a segmentation is equivalent to a random walk, optimality is defined as the minimization of the conditional Kullback-Leibler (KL) divergence between the target probabilities $P_{ij}^*$ and the transition probabilities $P_{ij}$ obtained by normalizing $S_{ij}$. Because $P^*$ is fixed, the above minimization is equivalent to maximizing the *cross entropy* between the two (conditional) distributions, i.e. max $J$, where

$$J = \sum_{i \in I} \frac{1}{|I|} \sum_{j \in I} P_{ij}^* \log P_{ij} \quad (9)$$

If we interpret the factor $1/|I|$ as a uniform distribution over states $\pi^0$ then the criterion in (9) is equivalent to the KL divergence between two distributions over transitions $KL(P_{i \to j}^{*1} || P_{i \to j})$ where $P_{i \to j}^{(*)} = \pi_i^0 P_{ij}^{(*)}$.

Maximizing $J$ can be done via gradient ascent in the parameters $\lambda$. We obtain

$$\frac{\partial J}{\partial \lambda_q} = \frac{1}{|I|} \left( \sum_{ij} P_{ij}^* f_{ij}^q - \sum_{ij} P_{ij} f_{ij}^q \right) \quad (10)$$

One can further note that the optimum of $J$ corresponds to the solution of the following maximum entropy problem:

$$\max_{P_{j|i}} H(j|i) \quad \text{s.t.} \quad < f_{ij}^q >_{\pi^0 P_{j|i}} = < f_{ij}^q >_{\pi^0 P_{j|i}^*} \text{ for } q = 1, \ldots Q \quad (11)$$

Since this is a convex optimization problem, it has a unique optimum.

# 6 Segmentation with shape and region information

In this section, we exemplify our approach on a set of synthetic and real images and we use features carrying contour and shape information. First we use a set of local filer banks as edge detectors. They capture both edge strength and orientation. From this basic information we construct two features: the *intervening contour* (IC) and the *co-linearity/co-circularity* (CL).

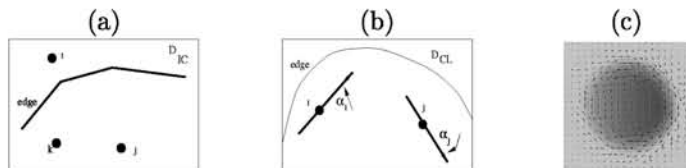

Figure 2: Features for segmenting objects with smooth rounded shape. (a) The edge strength provides a cue of region boundary. It biases against random walks in a direction orthogonal to an edge. (b) Edge orientation provides a cue for the object's shape. The induced edge flow is used to bias the random walk along the edge, and transitions between co-circular edge flows are encouraged. (c) Edge flow for the bump in figure 3.

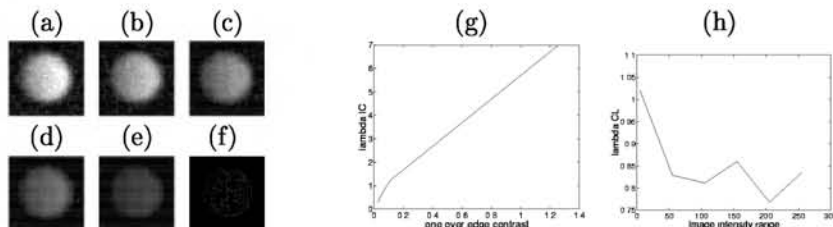

Figure 3: "Bump" images (a)-(f) with gradually reduced contrast are used for training. (g) shows the relation between the image edge contrast and the learned value of $\lambda_{IC}$, demonstrating automatic adaptation to the dynamic range of the IC. (h) shows the dependence on image contrast of $\lambda_{CL}$. At low image contrast, CL becomes more important.

The first feature is based on the assumption that if two pixels are separated by an edge, then they are less likely to belong together(figure 2). In the random walk interpretation, we are less likely to walk in a direction perpendicular to an edge. The intervening contour[6] is computed by $f_{ij}^{IC} = \text{MAX}_{k \in l(i,j)} \text{Edge}(k)$, where $l(i,j)$ is a line connecting pixel $i$ and $j$, and $\text{Edge}(k)$ is the edge strength at pixel $k$.

While the IC provides a cue for region boundaries, the edge orientation provides a cue for object shape. Human visual studies suggest that the shape of an object's boundary has a strong influence on how objects are grouped. For example, a convex region is more likely to be perceived as a single object Thinking of segmentation as a random walk provides a natural way of exploiting this knowledge. Each discrete edge in the image induces an *edge flow* in its neighborhood. To favor convex regions, we can further bias the random walk by enhancing the transition probabilities between pixels with co-circular edge flow. Thus we define the CL feature as: $f_{ij}^{CL} = \frac{2 - cos(2\alpha_i) - cos(2\alpha_j)}{1 - cos(\alpha_l)} + \frac{2 - cos(2\alpha_i + \alpha_j)}{1 - cos(\alpha_o)}$, where $\alpha_i, \alpha_j$ are defined as in figure 2(b).

For training, we have constructed the set of "bump" images with varying image contrast, as shown in figure 3. Figure 4 shows segmentation results using the weights trained with the "bump" image in figure 3(c).

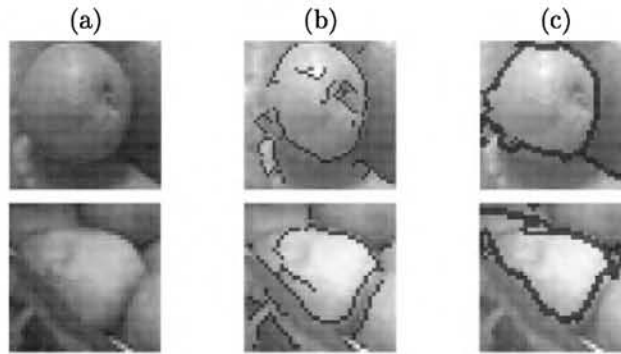

Figure 4: Testing on real images: (a) test images; (b) canny edges computed with the Matlab "edge" function; (c) NCut segmentation computed using the weights learned on the image in 5(c). The system learns to prefer contiguous groups with smooth boundary. The canny edge map indicates that simply looking for edges is likely gives brittle and less meaningful segmentations.

## 7  Conclusion

The main contribution of our paper is showing that spectral segmentation methods have a probabilistic foundation. In the framework of random walks, we give a new interpretation to the NCut criterion and algorithm and a better understanding of its motivation. The probabilistic framework also allows us to define a principled criterion for supervised learning of image segmentation.

**Acknowledgment**: J.S. is supported by DARPA N00014-00-1-0915, NSF IRI-9817496.

## References

[1] Fan R. K. Chung. *Spectral Graph Theory*. American Methematical Society, 1997.

[2] Ravi Kannan, Santosh Vempala, and Adrian Vetta. On clusterings: good, bad and spectral. In *Proc. 41st Symposium on the Foundations of Computer Science*, 2000.

[3] J. R. Kemeny and J. L. Snell. *Finite Markov Chains*. Van Nostrand, New York, 1960.

[4] Jon M. Kleinberg. Authoritative sources in a hyperlinked environment. Technical report, IBM Research Division, Almaden Research Center, 1997.

[5] M. Maila and J. Shi. A random walks view of spectral segmentation. In *Proc. International Workshop on AI and Statistics(AISTATS)*, 2001.

[6] Jitendra Malik, Serge Belongie, Thomas Leung, and Jianbo Shi. Contour and texture analysis for image segmentation. *International Journal of Computer Vision*, 2000.

[7] P. Perona and W. Freeman. A factorization approach to grouping. In *European Conference on Computer Vision*, 1998.

[8] G.L. Scott and H. C. Longuet-Higgins. Feature grouping by relocalsation of eigenvectors of the proximity matrix. In *Proc. British Machine Vision Conference*, 1990.

[9] J. Shi and J. Malik. Normalized cuts and image segmentation. *IEEE Transactions on Pattern Analysis and Machine Intelligence*, 2000. An earlier version appeared in CVPR 1997.

[10] Y. Weiss. Segmentation using eigenvectors: a unifying view. In *International Conference on Computer Vision*, 1999.
